# Grouping Contours Via a Related Image

**Praveen Srinivasan**
GRASP Laboratory
University of Pennsylvania
Philadelphia, PA 19104
psrin@seas.upenn.edu

**Liming Wang**
Fudan University
Shanghai, PRC 200433
wanglm@fudan.edu.cn

**Jianbo Shi**
GRASP Laboratory
University of Pennsylvania
Philadelphia, PA 19104
jshi@cis.upenn.edu

## Abstract

Contours have been established in the biological and computer vision literature as a compact yet descriptive representation of object shape. While individual contours provide structure, they lack the large spatial support of region segments (which lack internal structure). We present a method for further grouping of contours in an image using their relationship to the contours of a second, related image. Stereo, motion, and similarity all provide cues that can aid this task; contours that have similar transformations relating them to their matching contours in the second image likely belong to a single group. To find matches for contours, we rely only on shape, which applies directly to all three modalities without modification, in contrast to the specialized approaches developed for each independently. Visually salient contours are extracted in each image, along with a set of candidate transformations for aligning subsets of them. For each transformation, groups of contours with matching shape across the two images are identified to provide a context for evaluating matches of individual contour points across the images. The resulting contexts of contours are used to perform a final grouping on contours in the original image while simultaneously finding matches in the related image, again by shape matching. We demonstrate grouping results on image pairs consisting of stereo, motion, and similar images. Our method also produces qualitatively better results against a baseline method that does not use the inferred contexts.

## 1 Introduction

Researchers in biological vision have long hypothesized that image contours (ordered sets of edge pixels, or contour points) are a compact yet descriptive representation of object shape. In computer vision, there has been substantial interest in extracting contours from images as well as using object models based on contours for object recognition ([15, 5]), and 3D image interpretation [11].

We examine the problem of grouping contours in a single image aided by a related image, such as stereo pair, a frame from the same motion sequence, or a similar image. Relative motion of contours in one image to their matching contours in the other provides a cue for grouping. The contours themselves are detected bottom-up without a model, and are provided as input to our method. While contours already represent groupings of edges, they typically lack large spatial support. Region segments, on the other hand, have large spatial support, but lack the structure that contours provide. Therefore, additional grouping of contours can give us both qualities. This has important applications for object recognition and scene understanding, since these larger groups of contours are often large pieces of objects.

Figure 1 shows a single image in the 1st column, with contours; in the other columns, top row, are different images related by stereo, motion and similarity to the first, shown with their contours. Below each of these images are idealized groupings of contours in the original image. Note that internal contours on cars and buildings are grouped, providing rich, structured shape information over a larger image region.

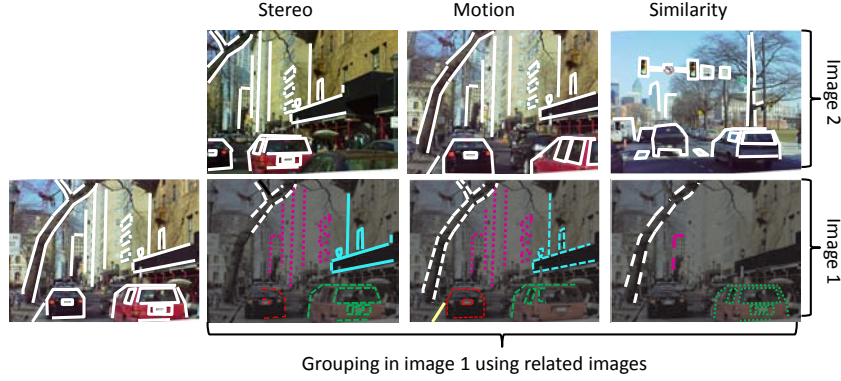

Figure 1: Contours (white) in the image on the left can be further grouped using the contours of a second, related image (top row). The bottom row shows idealized groupings in the original image according to the inter-image relationship.

## 2  Related Work

Stereo, motion, and similar image matching have been studied largely in isolation, and often with different purposes in mind than perceptual grouping. Much of the stereo literature focuses on per-pixel depth recovery; however, as [7] noted, stereo can be used for perceptual grouping without requiring precise depth estimation. Motion is often used for estimating optical flow or dense segmentation of images into groups of pixels undergoing similar motion [13]. These approaches to motion and stereo are largely region-based, and therefore do not provide the same internal structure that groups of contours provide. Similar image matching has been used for object recognition [1], but is rarely applied to image segmentation.

In work on contours, [12] matched contour points in the context of aerial imagery, but use constraints such as ordering of matches along scanlines that are not appropriate for motion or similar images, and do not provide grouping information. [9] grouped image pixels into contours according to similar motion using optical flow as a local cue. While the result addresses the long-standing aperture problem , it does not extend to large inter-image deformations or matching similar images. [8] grouped and matched image regions across different images and unstable segmentations (as we do with contours), but the regions lack internal structure. [2, 6] used stereo pairs of images to detect depth discontinuities as potential object boundaries. However, these methods will not detect and group group contours in the interior of fronto-parallel surfaces.

## 3  Grouping Criteria

We present definitions and basic criteria for grouping contours. The inputs to our method are:

1. Images: $\mathbf{I_1}$ $\mathbf{I_2}$; for each image $\mathbf{I_i}$ $i$  1 2 we also have:

2. A set of points (typically image edges) $I_i^P$; $p$  $I_i^P, p$  $R^2$. We restrict the set of points to those that lie on image contours, defined next.

3. A set of contours $I_i^C$, where the $j$th contour $C_i^j$  $I_i^C$ is an ordered subset of points in $I_i^P$: $C_i^j = [p_{k_1}\ p_{k_2}\quad p_{k_n}]$  $I_i^C$.

We would like to infer groups $\mathbf{G_1}$  $\mathbf{G_{nGroups}}$, each with the following attributes:

1. A transformation $T_i$ that aligns a subset of contours (e.g., corresponding to an object) in $\mathbf{I_1}$ to $\mathbf{I_2}$. $\mathbf{T}$ is the set of all $T_i$.

2. A subset of contours $\mathrm{Con}_i$ in each image, known as a **context**, such that the two subsets have similar overall shape. $\mathrm{Con}_i =$  $\mathrm{Con}_i^1$ $\mathrm{Con}_i^2$ ; $\mathrm{Con}_i^1$  0 1 $^{I_1^C}$ , $\mathrm{Con}_i^2$  0 1 $^{I_2^C}$ . Each $\mathrm{Con}_i^j$ is vector that indicates which contours are in the context for image $\mathbf{I_j}$. $\mathbf{Con}$ is the set of all $\mathrm{Con}_i$.

We further define the following variables on contours $C_1^j = [p_{k_1}\quad p_{k_n}]$:

1. A group label $l_j$; $l_j = a$ implies that $C_1^j$ belongs to group $\mathbf{G_a}$. $\mathbf{L} =$  $l_j$ , set of all labels.

2. Matches $\mathrm{Match}_j = [q_{r_1}\quad q_{r_n}]$, $q_{r_i}$  $I_2^P$, s.t. $p_{k_1}$ matches $q_{r_n}$. $\mathbf{Match} =$  $\mathrm{Match}_j$ , the set of all matches for each contour.

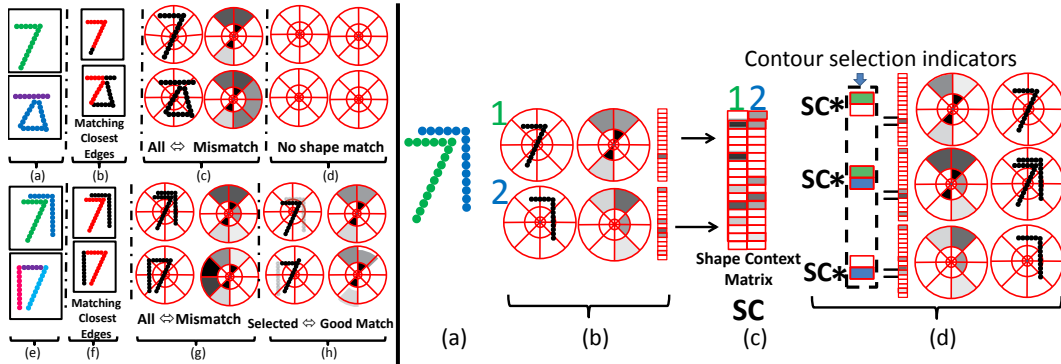

Figure 2: (left) - matching closest points cannot reject false positives; simply enlarging the feature size rejects true positives; increasing the feature size and selecting correct context fixes the problem and Figure 3: (right); the realized shape context from choosing a subset of contours can be summarized as the multiplication of a shape context matrix **M** and a binary indicator vector.

We would like the groups to possess the following criteria:

1. Good continuation - for contours that overlap significantly, we prefer that they are present in the same group, if they are grouped at all.

2. Common fate by shape: contours with similar transformations mapping them to their matches should be grouped. We also require that each contour point in a grouped contour and its matching point in the second image have similar local **shape**. Shape in each image is defined with respect to a subset of image contours known as a **context**; we will explain the importance of choosing the correct context for shape comparison, as first noted in [16].

3. Maximality/simplicity: We would like to group as many of the contours as possible into as few groups as possible, while still maintaining the similarity of local shape described above.

We will encode these criteria in our cost function $\mathbf{F}(\mathbf{T}\ \mathbf{Con}\ \mathbf{L}\ \mathbf{Match})$, which we seek to minimize. Our cost function has the following properties, which we develop in the following sections: 1) For fixed contexts **Con** and transformations $\mathbf{T}$, $\min_{\mathbf{Match}\ \mathbf{L}} \mathbf{F}(\mathbf{T}\ \mathbf{Con}\ \mathbf{L}\ \mathbf{Match})$ is a Markov random field (MRF) that can be minimized exactly via graph cuts ([3]). This corresponds to a standard computational formulation for graph matching (in this case, there is one graph in each image, over the contours). 2) For fixed matches **Match**, transformations, $\mathbf{T}$ and labels $\mathbf{L}$, $\mathbf{F}$ decomposes as the sum over $i = 1$ nGroups and we can minimize independently: $\min_{\mathbf{Con_i}} \mathbf{F_i}(\mathrm{T_i}\ \mathrm{Con_i}\ \mathrm{Match_{j\ l(j)=i}})$ as an integer linear program. This can be easily relaxed to a closely related linear program (LP), allowing for an efficient approximation. This combination of the MRF standard graph matching technique with an LP for inferring context for accurate matching by shape is our main contribution.

The layout of our paper is as follows: we explain the problem and importance of selecting contours as context for accurate matching and grouping, outline our computational solution (LP) for inferring **Con** given $\mathbf{T}$, our technique for choosing $\mathbf{T}$, followed by finding $\mathbf{L}$ and matches **Match** based on the inferred contexts (via graph cuts). Results using our method follow, and we demonstrate improvement over a baseline that lacks that benefits of our context selection procedure.

## 4   Matching and Context Selection

We can evaluate the hypothesis of a particular contour point match by comparing the local shape around the point and around its match. Although local features such as curvature and simple proximity (in the case of roughly aligned contours) have been used for matching ([12]), inconsistencies in the input contours across two images make these them prone to error. Local features exhibit completeness (good score for correctly matching shapes), but not soundness (bad score to not matching shapes). Figure 2 illustrates this distinction. Two aligned sets of contours are shown in a),e). In a), the contours do not match, while in e), a "7" shape is common to both. In b) and f), matching of

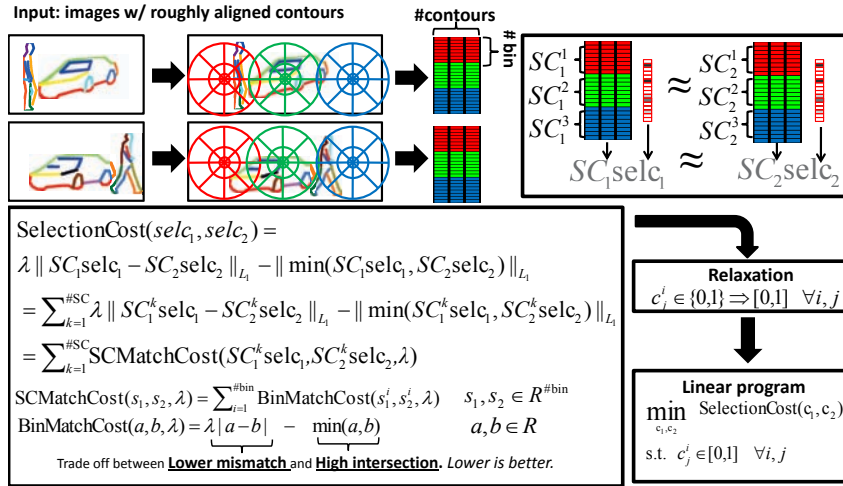

Figure 4: The context selection process for roughly aligned sets of contours. See text for full description.

closest points between two roughly aligned shapes finds matches in both examples due to the small support of local features, even though there is no valid match in a).

However, increasing the support of the feature does not solve the problem. As an example, we use the shape context, an image feature that has been widely used for shape matching ([1]). Briefly, a shape context provides a log-polar spatial histogram that records the number of points that fall into a particular bin. In Figure 2 c,g), shape contexts (darker bins mean larger bin count) with large spatial support placed according to the rough alignment exhibit high dissimilarity in both cases, failing to find a match in a). The large feature failed because contours in each image that had no match in the other image were used in computing the shape context. Inferring which contours to include and which to omit would give better features, as in Figure 2 d),h). This fixes the completeness problem, while retaining soundness: no combination of contours in the two images in a) can produce matching shapes. Therefore, with rough alignment we can cast the first step of shape matching as **context selection**: which subset of contours, or context, to use for feature computation. Given the correct context, matching individual contour points is much easier.

### 4.1 Selection Problem

We can neatly summarize the effect of a particular context selection on a shape context as seen in Figure 3. a) shows two contours which we can select from. b) shows the shape contexts for each individual contour. The bin counts for a shape context can be encoded as a vector, represented by the shape contexts and their vector representations alongside. In c), we put the vector form of these shape contexts into a matrix $SC$, where each column corresponds to one contour. $SC$ has dimensions nBins by nContours, where nBins is the number of bins in the shape context (and the length of the associated vector). The entry $SC(i \; j)$ is the bin count for bin $i$ and contour $j$. For each contour $C_i^j$ in an image $\mathbf{I_i}$, we associate an selection indicator variable $\mathrm{sel}_i^j \quad 0 \; 1$ , which indicates whether or not the contour is selected; the vector of these indicator variables is $\mathrm{sel}_i$. Then the shape context bin counts realized by a particular selection of contours is $SC\mathrm{sel}_i$, simply the multiplication of the matrix $SC$ and the vector $\mathrm{sel}_i$. d) shows the effect on the shape context histogram of various context selections.

### 4.2 Shape context matching cost

The effectiveness of selection depends significantly on the shape context matching cost. Traditional matching costs (Chi-square, $L_1$, $L_2$) only measure similarity, but selecting no contours in either image gives a perfect matching cost, since in both shape contexts, all bins will be 0. While similarity is important, so is including more contours rather than fewer (maximality).

Our shape context matching cost, $\mathrm{SCMatchCost}(s_1 \; s_2 \quad)$ in Figure 4, is a linear combination of the $L_1$ distance between shape context vectors $s_1$ and $s_2$ (similarity), and the intersection distance (maximality, one of our original grouping criteria), the $L_1$ norm of $\min(s_1 \; s_2)$ where $\min$ is element-

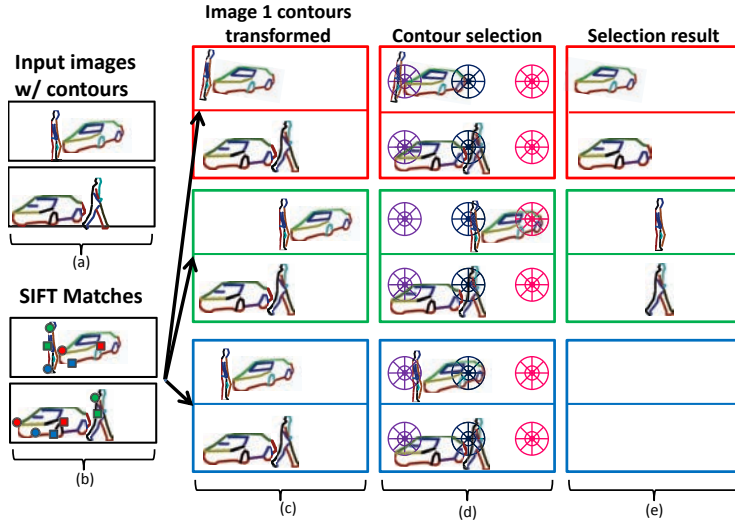

Figure 5: For a pair of images, SIFT matches propose different transformations of the contours in image 1 to align with contours in image 2. The selection process is run for each transformation to infer a context suitable for evaluating contour point matches via shape.

wise. The intersection term encourages higher bin counts in each shape context and therefore the inclusion of more contours. The parameter trades off between similarity and maximality; typically 1.

### 4.3 Computational Solution

Our formulation of the problem follows the construction first presented in [16], which studied the role of context selection for object recognition. Figure 4 shows the formulation of the overall selection cost SelectionCost. This minimizes $\mathbf{F_i}(T_i \; \mathrm{Con}_i \; \mathrm{Match}_{j \; l(j)=i})$ over $\mathrm{Con}_i$. We begin with two input images, where the contours are in rough alignment (by applying known $T_i$ to $I_1^C$). Multiple shape contexts are placed in each image on a uniform grid (an approximation of $\mathrm{Match}_{j \; l(j)=i}$, since we initially have no matches). Like-colored (in the figure) shape contexts will be compared across images. Our goal is to select contours in each image to minimize the sum of SCMatchCost for each pair of shape contexts. For each shape context $j$ in each image $i$, we compute the corresponding shape context matrix $SC_i^j$. All the $SC_i^j$ in a particular image $\mathbf{I_i}$ are stacked to form matrix $SC_i$. $SC_i$ for each image has been color coded to show the $SC_i^j$ matrix corresponding to each shape context.

We introduce the indicator vectors $\mathrm{selc}_1 = [\mathrm{selc}_1^1 \; \mathrm{selc}_1^m]$ and $\mathrm{selc}_2 = [\mathrm{selc}_1^1 \; \mathrm{selc}_1^n]$ for images $\mathbf{I_1} \; \mathbf{I_2}$. $\mathrm{selc}_i^j = 1$ implies that contour $C_i^j$ is selected. $SC_i\mathrm{selc}_i$ is then the realized bin counts for all the shape contexts in image $\mathbf{I_i}$ under selection $\mathrm{selc}_i$. We seek to choose $\mathrm{selc}_1$ and $\mathrm{selc}_2$ such that $SC_1\mathrm{selc}_1 \quad SC_2\mathrm{selc}_2$ in a shape sense; entries of $SC_1\mathrm{selc}_1$ and $SC_2\mathrm{selc}_2$, or realized bin counts, are in correspondence, so we can score these pairs of bin counts using BinMatchCost. A compact summary of this cost function SelectionCost is shown in Figure 4; its decomposition as the sum of SCMatchCost terms, which are each in turn a sum over BinMatchCost terms is shown.

The minimization of SelectionCost over $\mathrm{selc}_1$ and $\mathrm{selc}_2$ is in fact an integer linear program ($L_1$ distance and $\min$ are easily encoded with additional variables and linear constraints). By relaxing each $\mathrm{sel}_i^j \quad 0 \; 1 \quad [0 \; 1]$, we obtain a linear program (LP) which can be solved efficiently using standard solvers (e.g. SDPT3). Although other methods exist for solving integer linear programs, such as branch-and-bound, we found that directly discretizing the $\mathrm{sel}_i^j$ with a fixed threshold worked well. Then $\mathrm{Con}_i = \mathrm{selc}_1 \; \mathrm{selc}_2$.

### 4.4 Multiple Context Selections for Image Matching

Now that we have established how to do selection in the case were are given $T_i$, we now apply it in images where there may be multiple objects that are related across the two images by different alignments. We first need to infer the set of candidate transformations $\mathbf{T}$; for our purposes, we will

restrict them to be similarity transforms, although we note that non-linear or piecewise linear (e.g., articulation) transformations could certainly be used. A simple method for proposing transformations in the two images is via SIFT ([10]) feature matches. A SIFT match provides scale, orientation, and translation (a similarity transform). RANSAC with multiple matches can be used to estimate full homographies, similar to [14].

Figure 5 depicts an idealized selection process for two images (only the contours are shown). For groups of SIFT matches that describe similar transformations, a transformation $T_i$ is extracted and warps the contours in image 1 to line up with those of image 2, in c). The selection problem is formulated separately for each set of aligned contours d). The solution vectors of the $\mathrm{SelectionCost}$ LP for each $T_i$ provide a context $\{\widehat{\mathrm{Con}_i^1}, \mathrm{Con}_i^2\}$ ($\{\mathrm{selc}_1, \mathrm{selc}_2\}$ previously) of matching contours, e). Two correct transforms align the car and person, and the selection result includes the respective contours (rows 1,2 of e). A third, wrong transform results in an empty selection (row 3 of e). We can view the context selection procedure for minimizing $F_i$ as choosing the context of contours so as to best reduce the matching cost of the hypothesized inter-image matches for contours with label $i$, under the transformation $\mathrm{T}_i$. In a sense, we are optimizing the local features via an LP, which traditional graph matching techniques do not do. The result of this optimization will appear in the unary term of the label/match MRF described next.

## 5    Graph Cuts for Group Assignment and Matching

We previously computed context selections (as solutions to the $\mathrm{SelectionCost}$ LP), which found groups of contours in each image that have similar shape, $\widehat{\mathbf{Con}} = \{\widehat{\mathrm{Con}_1}, ..., \widehat{\mathrm{Con}_{\mathrm{nGroups}}}\}$ under transformations $\widehat{\mathbf{T}}$. Given these, we seek to compute $\mathbf{L}$ and $\mathbf{Match}$. Some labels in $1, ..., \mathrm{nGroups}$ may not be assigned to any contours, satisfying our simplicity criterion for grouping. Note that a contour $C_1^j$ need not be selected as context in a particular group $a$ in order to have $l_j = a$. Recall with respect to the original cost function, we seek to optimize: $\min_{\mathbf{Match}, \mathbf{L}} \mathbf{F}(\widehat{\mathbf{T}}, \widehat{\mathbf{Con}}, \mathbf{L}, \mathbf{Match})$ We phrase this label assignment problem as inference in a Markov network (MN). The MN encodes the joint distribution over the labels $L$ as a product of potentials: $P(L) = \frac{1}{Z} \prod_j \phi(l_j) \prod_{j,k} \phi(l_j, l_k)$ where $Z$ is a normalization constant.

The binary potentials $\phi(l_j, l_k)$ encode the preference that overlapping contours $C_1^j, C_1^k$ have the same label:

$$\phi(l_j = a, l_k = b) = \begin{cases} 1 & a = b \\ 1 - \tau & a \neq b \end{cases} \qquad (1)$$

where $0 \leq \tau \leq 1$ controls the penalty of having different labels. This is a simple smoothing potential to encourage continuity. Two contours overlap if they contain at least one point in common.

The unary potential $\phi(l_j)$ encodes how well contour $C_1^j = [p_{k_1}, p_{k_2}, ..., p_{k_n}]$ can be matched in the second image with respect to the context $\{\widehat{\mathrm{Con}_a^1}, \mathrm{Con}_a^2\}$. The log-unary potential decomposes as the sum of matching costs of the individual points $p_{k_i}$ to their best match in image $\mathbf{I_2}$, with respect to the context $\{\widehat{\mathrm{Con}_a^1}, \mathrm{Con}_a^2\}$:

$$\log \phi(l_j = a) \propto -\sum_{i=1}^{n} [\min_{q \in I_2^P} \mathrm{MatchCostInContext}(p_{k_i}, q, a)] \qquad (2)$$

where $\mathrm{MatchCostInContext}(p, q, a) = \mathrm{SCMatchCost}(SC_1^{T_a(p)} \widehat{\mathrm{Con}_a^1}, SC_2^q \widehat{\mathrm{Con}_a^2})$ and $SC_1^p$ and $SC_2^q$ are respectively the shape context matrix computed for a shape context centered at $T_a(p)$ using the contours in image 1 under transformation $T_a$, and the matrix for a shape context centered at $q$ using the contours in image 2.

We compute the exact MAP estimate in the MN using the $\alpha - \beta$ swap graph cut algorithm ([3]), which can maximize this type of energy. Instead of using all contours image 1 as nodes in the MN, we only allow contours were selected in at least one of the context $\mathrm{Con}_i^1$; likewise, we only permit matches to points in image 2 that appear in a contour selected in at least one $\mathrm{Con}_j^2$. This better allows us to deal with contours that appear only in one image and thus cannot be reliably grouped based on relative motion.

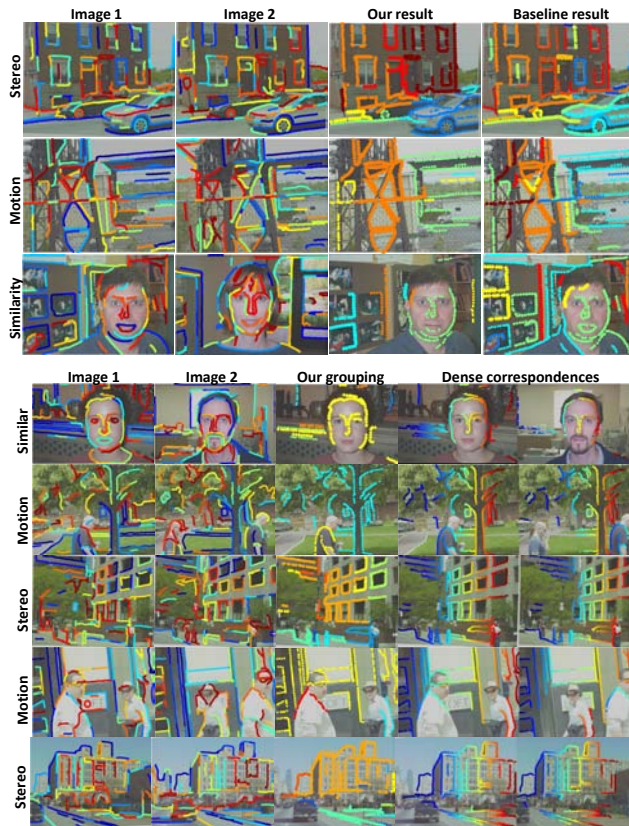

Figure 6: Baseline comparison (top) and additional results (bottom). Top: Columns 1,2: original images with input contours, each colored. Columns 3,4: grouping results for our method and baseline; groups of contours are a single color. In stereo pairs, like colors indicate similar disparity. Bottom: Columns 1,2: original images with input contours, each colored. Column 3: our grouping result. Columns 4,5: matches across images indicated by like colors. Please view in color.

### 5.1  Baseline Comparison

As a baseline comparison, we attempted grouping using an MN that involved no selection information. The binary potential remained the same, while the unary potential $\phi(l_j = a)$ was a function of the distance of each contour point in contour $C_1^j$ to its closest match in $I_2^P$, under the transformation $T_a$:

$$\log \phi(l_j = a) \propto -\sum_{i=1}^{n} [\min_{q \in I_2^P} (||T_a(p_{k_i}) - q||_{L2}^2, \text{occlusionThresh}^2)] \tag{3}$$

The constant occlusionThresh serves a threshold in case a contour point had no nearby match in $I_2^P$ under the transformation $T^a$. Points which had no match within occlusionThresh distance were marked as occluded for the hypothesis $l_j = a$. If more than half the points in the final assignment $l_j^*$ for a contour were occluded, we marked the entire contour as occluded, and it was not displayed. Since we omitted all selection information, all contours in the 1st image were included in the MN as nodes, and their contour points were allowed to match to any contour point in $I_2^P$. We again optimized the MN energy with the $\alpha - \beta$ swap graph cut. Free parameters were tuned by hand to produce the best result possible.

## 6  Experiments

We tested our method and the baseline over stereo, motion and similar image pairs. Input contours in each image were extracted automatically using the method of [15]. SIFT matches were extracted from images, keeping only confident matches as described in [10]; matches proposing similar transformations were pruned to a small set, typically 10-20. Because of the high quality of the inferred contexts, we used large shape contexts (radius 90 pixels, in images of size ≈ 400 by ≈ 500), which

made matching very robust. The shape contexts were augmented with edge orientation bins in addition to the standard radial and angular bins. Shape contexts were placed on a uniform grid atop the registered contours (via $T_i$) with a spacing 50 pixels in the $x$ and $y$ dimensions. Image pairs were taken from the Caltech 101 dataset [4] and from a stereo rig with 1m baseline mounted on a car from our lab (providing stereo and motion images). The running time of our unoptimized MATLAB implementation was several minutes for each image pair.

Figure 6, top block, shows the results of our method and the baseline method on stereo, motion and similar images. We can see that our method provides superior groupings that better respect object boundaries. Groups for stereo image pairs are colored according to disparity. Due to the lack of large context, the baseline method is able to find a good match for a given contour point under almost any group hypothesis $l_j = a$, since in cluttered regions, there are always nearby matches. However, by using a much larger, optimized context, our method exploits large-scale shape information and is better able to infer about occlusion, as well as layer assignment. We present additional results on different images in Figure 6, bottom block, and also show the dense correspondences. Interesting groups found in our results include facades of buildings, people, and a car (top row).

## 7  Conclusion

We introduced the problem of grouping of contours in an image using a related image, such as stereo, motion or similar, as an important step for object recognition and scene understanding. Grouping depends on the ability to match contours across images to determine their relative motion. Selecting a good context for shape evaluation was key to robust simultaneous and grouping of contours across images. A baseline method similar to our proposed method, but without context, produced worse groupings on stereo, motion and similar images. Future work will include trying to learn 3D object models from stereo and motion images, and a probabilistic formulation of the matching framework. Introducing learning to improve the grouping result is also an area of significant interest; some shape configurations are more reliable for matching than others.

## References

[1] S. Belongie, J. Malik, and J. Puzicha. Shape matching and object recognition using shape contexts. *IEEE PAMI*, 2002.

[2] S. Birchfield and C. Tomasi. Depth discontinuities by pixel-to-pixel stereo. In *ICCV*, 1998.

[3] Y. Boykov, O. Veksler, and R. Zabih. Fast approximate energy minimization via graph cuts. *PAMI*, 2001.

[4] L. Fei-Fei, R. Fergus, and P. Perona. One-shot learning of object categories. *PAMI*, 2006.

[5] V. Ferrari, L. Fevrier, F. Jurie, and C. Schmid. Groups of adjacent contour segments for object detection. *PAMI*, 2008.

[6] M. Gelautz and D. Markovic. Recognition of object contours from stereo images: an edge combination approach. *3D PVT*, 2004.

[7] W.E.L. Grimson. Why stereo vision is not always about 3d reconstruction. In *MIT AI Memo, Technical Report AIM-1435*, 1993.

[8] V. Hedau, H. Arora, and N. Ahuja. Matching images under unstable segmentation. In *CVPR 2008*.

[9] C. Liu, W. T. Freeman, and E.H. Adelson. Analysis of contour motions. In *NIPS*, 2006.

[10] D. Lowe. Distinctive image features from scale-invariant keypoints. In *IJCV*, 2003.

[11] D.G. Lowe and T.O. Binford. The recovery of three-dimensional structure from image curves. *PAMI*, 1985.

[12] D. Sherman and S. Peleg. Stereo by incremental matching of contours. *PAMI*, 1990.

[13] J.Y.A. Wang and E.H. Adelson. Layered representation for motion analysis. In *CVPR*, 1993.

[14] J. Wills, S. Agarwal, and S. Belongie. A feature-based approach for dense segmentation and estimation of large disparity motion. *IJCV*, 2006.

[15] Q. Zhu, G. Song, and J. Shi. Untangling cycles for contour grouping. In *ICCV 2007*.

[16] Qihui Zhu, Liming Wang, Yang Wu, and Jianbo Shi. Contour context selection for object detection: A set-to-set contour matching approach. In *ECCV*, 2008.

